# Learning Label Embeddings for Nearest-Neighbor Multi-class Classification with an Application to Speech Recognition

**Natasha Singh-Miller**
Massachusetts Institute of Technology
Cambridge, MA
natashas@mit.edu

**Michael Collins**
Massachusetts Institute of Technology
Cambridge, MA
mcollins@csail.mit.edu

## Abstract

We consider the problem of using nearest neighbor methods to provide a conditional probability estimate, $P(y|\mathbf{a})$, when the number of labels $y$ is large and the labels share some underlying structure. We propose a method for learning label embeddings (similar to error-correcting output codes (ECOCs)) to model the similarity between labels within a nearest neighbor framework. The learned ECOCs and nearest neighbor information are used to provide conditional probability estimates. We apply these estimates to the problem of acoustic modeling for speech recognition. We demonstrate significant improvements in terms of word error rate (WER) on a lecture recognition task over a state-of-the-art baseline GMM model.

## 1 Introduction

Recent work has focused on the learning of similarity metrics within the context of nearest-neighbor (NN) classification [7, 8, 12, 15]. These approaches learn an embedding (for example a linear projection) of input points, and give significant improvements in the performance of NN classifiers.

In this paper we focus on the application of NN methods to multi-class problems, where the number of possible labels is large, and where there is significant structure within the space of possible labels. We describe an approach that induces prototype vectors $\mathbf{M}_y \in \Re^L$ (similar to error-correcting output codes (ECOCs)) for each label $y$, from a set of training examples $\{(\mathbf{a}_i, y_i)\}$ for $i = 1 \ldots N$. The prototype vectors are embedded within a NN model that estimates $P(y|\mathbf{a})$; the vectors are learned using a leave-one-out estimate of conditional log-likelihood (CLL) derived from the training examples. The end result is a method that embeds labels $y$ into $\Re^L$ in a way that significantly improves conditional log-likelihood estimates for multi-class problems under a NN classifier.

The application we focus on is acoustic modeling for speech recognition, where each input $\mathbf{a} \in \Re^D$ is a vector of measured acoustic features, and each label $y \in \mathcal{Y}$ is an acoustic-phonetic label. As is common in speech recognition applications, the size of the label space $\mathcal{Y}$ is large (in our experiments we have 1871 possible labels), and there is significant structure within the labels: many acoustic-phonetic labels are highly correlated or confusable, and many share underlying phonological features. We describe experiments measuring both conditional log-likelihood of test data, and word error rates when the method is incorporated within a full speech recogniser. In both settings the experiments show significant improvements for the ECOC method over both baseline NN methods (e.g., the method of [8]), as well as Gaussian mixture models (GMMs), as conventionally used in speech recognition systems.

While our experiments are on speech recognition, the method should be relevant to other domains which involve large multi-class problems with structured labels—for example problems in natural language processing, or in computer vision (e.g., see [14] for a recent use of neighborhood com-

ponents analysis (NCA) [8] within an object-recognition task with a very large number of object labels). We note also that the approach is relatively efficient: our model is trained on around 11 million training examples.

## 2  Related Work

Several pieces of recent work have considered the learning of feature space embeddings with the goal of optimizing the performance of nearest-neighbor classifiers [7, 8, 12, 15]. We make use of the formalism of [8] as the starting point in our work. The central contrast between our work and this previous work is that we learn an embedding of the *labels* in a multi-class problem; as we will see, this gives significant improvements in performance when nearest-neighbor methods are applied to multi-class problems arising in the context of speech recognition.

Our work is related to previous work on error-correcting output codes for multi-class problems. [1, 2, 4, 9] describe error-correcting output codes; more recently [2, 3, 11] have described algorithms for learning ECOCs. Our work differs from previous work in that ECOC codes are learned within a nearest-neighbor framework. Also, we learn the ECOC codes in order to model the underlying structure of the label space and not specifically to combine the results of multiple classifiers.

## 3  Background

The goal of our work is to derive a model that estimates $P(y|\mathbf{a})$ where $\mathbf{a} \in \Re^D$ is a feature vector representing some input, and $y$ is a label drawn from a set of possible labels $\mathcal{Y}$. The parameters of our model are estimated using training examples $\{(\mathbf{a}_1, y_1), ..., (\mathbf{a}_N, y_N)\}$. In general the training criterion will be closely related to the conditional log-likelihood of the training points:

$$\sum_{i=1}^{N} \log P(y_i|\mathbf{a}_i)$$

We choose to optimize the log-likelihood rather than simple classification error, because these estimates will be applied within a larger system, in our case a speech recognizer, where the probabilities will be propagated throughout the recognition model; hence it is important for the model to provide well-calibrated probability estimates.

For the speech recognition application considered in this paper, $\mathcal{Y}$ consists of 1871 acoustic-phonetic classes that may be highly correlated with one another. Leveraging structure in the label space will be crucial to providing good estimates of $P(y|\mathbf{a})$; we would like to learn the inherent structure of the label space automatically. Note in addition that efficiency is important within the speech recognition application: in our experiments we make use of around 11 million training samples, while the dimensionality of the data is $D = 50$.

In particular, we will develop nearest-neighbor methods that give an efficient estimate of $P(y|\mathbf{a})$. As a first baseline approach—and as a starting point for the methods we develop—consider the neighbor components analysis (NCA) method introduced by [8]. In NCA, for any test point $\mathbf{a}$, a distribution $\alpha(j|\mathbf{a})$ over the training examples is defined as follows where $\alpha(j|\mathbf{a})$ decreases rapidly as the distance between $\mathbf{a}$ and $\mathbf{a}_j$ increases.

$$\alpha(j|\mathbf{a}) = \frac{e^{-||\mathbf{a}-\mathbf{a}_j||^2}}{\sum_{m=1}^{N} e^{-||\mathbf{a}-\mathbf{a}_m||^2}} \tag{1}$$

The estimate of $P(y|\mathbf{a})$ is then defined as follows:

$$P_{nca}(y|\mathbf{a}) = \sum_{i=1, y_i=y}^{N} \alpha(i|\mathbf{a}) \tag{2}$$

In NCA the original training data consists of points $(\mathbf{x}_i, y_i)$ for $i = 1 \ldots N$, where $\mathbf{x}_i \in \Re^{D'}$, with $D'$ typically larger than $D$. The method learns a projection matrix $\mathbf{A}$ that defines the modified representation $\mathbf{a}_i = \mathbf{A}\mathbf{x}_i$ (the same transformation is applied to test points). The matrix $\mathbf{A}$ is learned from training examples, to optimize log-likelihood under the model in Eq. 2.

In our experiments we assume that $\mathbf{a} = \mathbf{A}\mathbf{x}$ for some underlying representation $\mathbf{x}$ and a projection matrix $\mathbf{A}$ that has been learned using NCA to optimize the log-likelihood of the training set. As a result the matrix $\mathbf{A}$, and consequently the representation $\mathbf{a}$, are well-calibrated in terms of using nearest neighbors to estimate $P(y|\mathbf{a})$ through Eq. 2. A first baseline method for our problem is therefore to directly use the estimates defined by Eq. 2.

We will, however, see that this baseline method performs poorly at providing estimates of $P(y|\mathbf{a})$ within the speech recognition application. Importantly, the model fails to exploit the underlying structure or correlations within the label space. For example, consider a test point that has many neighbors with the phonemic label /s/. This should be evidence that closely related phonemes, /sh/ for instance, should also get a relatively high probability under the model, but the model is unable to capture this effect.

As a second baseline, an alternative method for estimating $P(y|\mathbf{a})$ using nearest neighbor information is the following:

$$P_k(y|\mathbf{a}) = \frac{\text{\# of } k\text{-nearest neighbors of } \mathbf{a} \text{ in training set with label } y}{k}$$

Here the choice of $k$ is crucial. A small $k$ will be very sensitive to noise and necessarily lead to many classes receiving a probability of zero, which is undesirable for our application. On the other hand, if $k$ is too large, samples from far outside the neighborhood of $\mathbf{a}$ will influence $P_k(y|\mathbf{a})$. We will describe a baseline method that interpolates estimates from several different values of $k$. This baseline will be useful with our approach, but again suffers from the fact that it does not model the underlying structure of the label space.

## 4   Error-Correcting Output Codes for Nearest-Neighbor Classifiers

We now describe a model that uses error correcting output codes to explicitly represent and learn the underlying structure of the label space $\mathcal{Y}$. For each label $y$, we define $\mathbf{M}_y \in \Re^L$ to be a prototype vector. We assume that the inner product $\langle \mathbf{M}_y, \mathbf{M}_z \rangle$ will in some sense represent the similarity between labels $y$ and $z$. The vectors $\mathbf{M}_y$ will be learned automatically, effectively representing an embedding of the labels in $\Re^L$. In this section we first describe the structure of the model, and then describe a method for training the parameters of the model (i.e., learning the prototype vectors $\mathbf{M}_y$).

### 4.1   ECOC Model

The ECOC model is defined as follows. When considering a test sample $\mathbf{a}$, we first assign weights $\alpha(j|\mathbf{a})$ to points $\mathbf{a}_j$ from the training set through the NCA definition in Eq. 1. Let $\mathbf{M}$ be a matrix that contains all the prototype vectors $\mathbf{M}_y$ as its rows. We can then construct a vector $H(\mathbf{a}; \mathbf{M})$ that uses the weights $\alpha(j|\mathbf{a})$ and the true labels of the training samples to calculate the expected value of the output code representing $\mathbf{a}$.

$$H(\mathbf{a}; \mathbf{M}) = \sum_{j=1}^{N} \alpha(j|\mathbf{a}) \mathbf{M}_{y_j}$$

Given this definition of $H(\mathbf{a}; \mathbf{M})$, our estimate under the ECOC model is defined as follows:

$$P_{ecoc}(y|\mathbf{a}; \mathbf{M}) = \frac{e^{\langle \mathbf{M}_y, H(\mathbf{a}; \mathbf{M}) \rangle}}{\sum_{y' \in \mathcal{Y}} e^{\langle \mathbf{M}_{y'}, H(\mathbf{a}; \mathbf{M}) \rangle}}$$

| $L$ | average CLL |
|-----|-------------|
| 2   | -4.388      |
| 10  | -2.748      |
| 20  | -2.580      |
| 30  | -2.454      |
| 40  | -2.432      |
| 50  | -2.470      |
| 60  | -2.481      |

Table 1: Average CLL achieved by $P_{ecoc}$ over DevSet1 for different values of $L$

This distribution assigns most of the probability for a sample vector $\mathbf{a}$ to classes whose prototype vectors have a large inner product with $H(\mathbf{a}; \mathbf{M})$. All labels receive a non-zero weight under $P_{ecoc}(y|\mathbf{a}; \mathbf{M})$.

## 4.2 Training the ECOC Model

We now describe a method for estimating the ECOC vectors $\mathbf{M}_y$ in the model. As in [8] the method uses a leave-one-out optimization criterion, which is particularly convenient within nearest-neighbor approaches. The optimization problem will be to maximize the conditional log-likelihood function

$$F(\mathbf{M}) = \sum_{i=1}^{N} \log P_{ecoc}^{(loo)}(y_i|\mathbf{a}_i; \mathbf{M})$$

where $P_{ecoc}^{(loo)}(y_i|\mathbf{a}_i; \mathbf{M})$ is a leave-one-out estimate of the probability of label $y_i$ given the input $\mathbf{a}_i$, assuming an ECOC matrix $\mathbf{M}$. This criterion is related to the classification performance of the training data and also discourages the assignment of very low probability to the correct class.

The estimate $P_{ecoc}^{(loo)}(y_i|\mathbf{a}_i; \mathbf{M})$ is given through the following definitions:

$$\alpha^{(loo)}(j|i) = \frac{e^{-||\mathbf{a}_i - \mathbf{a}_j||^2}}{\sum_{m=1, m \neq i}^{N} e^{-||\mathbf{a}_i - \mathbf{a}_m||^2}} \text{ if } i \neq j \text{ and } 0 \text{ otherwise}$$

$$H^{(loo)}(\mathbf{a}_i; \mathbf{M}) = \sum_{j=1}^{N} \alpha^{(loo)}(j|i)\mathbf{M}_{y_j}$$

$$P_{ecoc}^{(loo)}(y|\mathbf{a}_i; \mathbf{M}) = \frac{e^{\langle \mathbf{M}_y, H^{(loo)}(\mathbf{a}; \mathbf{M}) \rangle}}{\sum_{y' \in \mathcal{Y}} e^{\langle \mathbf{M}_{y'}, H^{(loo)}(\mathbf{a}; \mathbf{M}) \rangle}}$$

The criterion $F(\mathbf{M})$ can be optimized using gradient-ascent methods, where the gradient is as follows:

$$\frac{\partial F(\mathbf{M})}{\partial \mathbf{M}_z} = \nabla(z) - \nabla'(z)$$

$$\nabla(z) = \sum_{i=1}^{N} \sum_{j=1}^{N} [\alpha^{(loo)}(j|i)(\delta_{z,y_i}\mathbf{M}_{y_j} + \delta_{y_j,z}\mathbf{M}_{y_i})]$$

$$\nabla'(z) = \sum_{i=1}^{N} \sum_{y' \in \mathcal{Y}} P_{ecoc}^{(loo)}(y'|\mathbf{a}_i; \mathbf{M}) \left( \sum_{j=1}^{N} [\alpha^{(loo)}(j|i)(\delta_{z,y'}\mathbf{M}_{y_j} + \delta_{y_j,z}\mathbf{M}_{y'})] \right)$$

| Model | Average CLL on DevSet 1 | Perplexity |
|---|---|---|
| $P_{nca}$ | -2.657 | 14.25 |
| $P_{nn}$ | -2.535 | 12.61 |
| $P_{ecoc}$ | -2.432 | 11.38 |
| $P_{full}$ | -2.337 | 10.35 |
| $P_{gmm}$ | -2.299 | 9.96 |
| $P_{mix}$ | -2.165 | 8.71 |

Table 2: Average conditional log-likelihood (CLL) of $P_{nca}$, $P_{nn}$, $P_{ecoc}$, $P_{nn'}$, $P_{gmm}$ and $P_{mix}$ on DevSet1. The corresponding perplexity values are indicated as well where the perplexity is defined as $e^{-x}$ given that $x$ is the average CLL.

Here $\delta_{a,b} = 1$ if $a = b$ and $\delta_{a,b} = 0$ if $a \neq b$. Since $\alpha^{(loo)}(j|i)$ will be very small if $||\mathbf{a}_i - \mathbf{a}_j||^2$ is large, the gradient calculation can be truncated for such pairs of points which significantly improves the efficiency of the method (a similar observation is used in [8]). This optimization is non-convex and it is possible to converge to a local optimum.

In our experiments we learn the matrix $\mathbf{M}$ using conjugate gradient ascent, though alternatives such as stochastic gradient can also be used. A random initialization of $\mathbf{M}$ is used for each experiment. We select $L = 40$ as the length of the prototype vectors $\mathbf{M}_y$. We experimented with different values of $L$. The average conditional log-likelihood achieved on a development set of approximately 115,000 samples (DevSet1) is listed in Table 1. The performance of the method improves initially as the size of $L$ increases, but the objective levels off around $L = 40$.

## 5 Experiments on Log-Likelihood

We test our approach on a large-vocabulary lecture recognition task [6]. This is a challenging task that consists of recognizing college lectures given by multiple speakers. We use the SUMMIT recognizer [5] that makes use of 1871 distinct class labels. The acoustic vectors we use are 112 dimensional vectors consisting of eight concatenated 14 dimensional vectors of MFCC measurements. These vectors are projected down to 50 dimensions using NCA as described in [13]. This section describes experiments comparing the ECOC model to several baseline models in terms of their performance on the conditional log-likelihood of sample acoustic vectors.

The baseline model, $P_{nn}$, makes use of estimates $P_k(y|\mathbf{a})$ as defined in section 3. The set $\mathcal{K}$ is a set of integers representing different values for $k$, the number of nearest neighbors used to evaluate $P_k$. Additionally, we assume $d$ functions over the the labels, $P_1(y), ..., P_d(y)$. (More information on the functions $P_j(y)$ that we use in our experiments can be found in the appendix. We have found these functions over the labels are useful within our speech recognition application.) The model is then defined as

$$P_{nn}(y|\mathbf{a}; \bar{\lambda}) = \sum_{k \in \mathcal{K}} \lambda_k P_k(y|\mathbf{a}) + \sum_{j=1}^{d} \lambda_j^0 P_j(y)$$

where $\lambda_k \geq 0, \forall k \in \mathcal{K}$, $\lambda_j^0 \geq 0$ for $j = 1, ..., d$, and $\sum_{k \in \mathcal{K}} \lambda_k + \sum_{j=1}^{d} \lambda_j^0 = 1$. The $\bar{\lambda}$ values were estimated using the EM algorithm on a validation set of examples (DevSet2). In our experiments, we select $\mathcal{K} = \{5, 10, 20, 30, 50, 100, 250, 500, 1000\}$. Table 2 contains the average conditional log-likelihood achieved on a development set (DevSet1) by $P_{nca}$, $P_{nn}$ and $P_{ecoc}$. These results show that $P_{ecoc}$ clearly outperforms these two baseline models.

In a second experiment we combined $P_{ecoc}$ with $P_{nn}$ to create a third model $P_{full}(y|\mathbf{a})$. This model includes information from the nearest neighbors, the output codes, as well as the distributions over the label space. The model takes the following form:

$$P_{full}(y|\mathbf{a}; \bar{\lambda}) = \sum_{k \in \mathcal{K}} \lambda_k P_k(y|\mathbf{a}) + \sum_{j=1}^{d} \lambda_j^0 P_j(y) + \lambda_{ecoc} P_{ecoc}(y|\mathbf{a}; \mathbf{M})$$

| Acoustic Model | WER (DevSet3) | WER (Test Set) |
|---|---|---|
| Baseline Model | 36.3 | 35.4 |
| Augmented Model | **35.2** | **34.5** |

Table 3: WER of recognizer for different acoustic models on the development and test set.

The values of $\bar{\lambda}$ here have similar constraints as before and are again optimized using the EM algorithm. Results in Table 2 show that this model gives a further clear improvement over $P_{ecoc}$.

We also compare ECOC to a GMM model, as conventionally used in speech recognition systems. The GMM we use is trained using state-of-the-art algorithms with the SUMMIT system [5]. The GMM defines a generative model $P_{gmm}(\mathbf{a}|y)$; we derive a conditional model as follows:

$$P_{gmm}(y|\mathbf{a}) = \frac{P_{gmm}(\mathbf{a}|y)^{\alpha}P(y)}{\sum_{y'\in\mathcal{Y}}P_{gmm}(\mathbf{a}|y')^{\alpha}P(y')}$$

The parameter $\alpha$ is selected experimentally to achieve maximum CLL on DevSet2 and $P(y)$ refers to the prior over the labels calculated directly from their relative proportions in the training set. Table 2 shows that $P_{full}$ and $P_{gmm}$ are close in performance, with $P_{gmm}$ giving slightly improved results. A final interpolated model with similar constraints on the values of $\bar{\lambda}$ trained using the EM algorithm is as follows:

$$P_{mix}(y|\mathbf{a};\bar{\lambda}) = \sum_{k\in\mathcal{K}}\lambda_k P_k(y|\mathbf{a}) + \sum_{j=1}^{d}\lambda_j^0 P_j(y) + \lambda_{ecoc}P_{ecoc}(y|\mathbf{a};\mathbf{M}) + \lambda_{gmm}P_{gmm}(y|\mathbf{a})$$

Results for $P_{mix}$ are shown in the final row in the table. This interpolated model gives a clear improvement over both the GMM and ECOC models alone. Thus the ECOC model, combined with additional nearest-neighbor information, can give a clear improvement over state-of-the-art GMMs on this task.

## 6 Recognition Experiments

In this section we describe experiments that integrate the ECOC model within a full speech recognition system. We learn parameters $\bar{\lambda}$ using both DevSet1 and DevSet2 for $P_{full}(y|\mathbf{a})$. However, we need to derive an estimate for $P(\mathbf{a}|y)$ for use by the recognizer. We can do so by using an estimate for $P(\mathbf{a}|y)$ proportional to $\frac{P(y|\mathbf{a})}{P(y)}$ [16]. The estimates for $P(y)$ are derived directly from the proportions of occurrences of each acoustic-phonetic class in the training set.

In our experiments we consider the following two methods for calculating the acoustic model.

- Baseline Model: $\beta_1 \log P_{gmm}(\mathbf{a}|y)$
- Augmented Model: $\beta_2 \log\left(\frac{\gamma P_{gmm}(y|\mathbf{a}) + (1-\gamma)P_{full}(y|\mathbf{a})}{P(y)}\right)$

The baseline method is just a GMM model with the commonly used scaling parameter $\beta_1$. The augmented model combines $P_{gmm}$ linearly with $P_{full}$ using parameter $\gamma$ and the log of the combination is scaled by parameter $\beta_2$. The parameters $\beta_1, \beta_2, \gamma$ are selected using the downhill simplex algorithm by optimizing WER over a development set [10]. Our development set (DevSet3) consists of eight hours of data including six speakers and our test set consists of eight hours of data including five speakers. Results for both methods on the development set and test set are presented in Table 3.

The augmented model outperforms the baseline GMM model. This indicates that the nearest neighbor information along with the ECOC embedding, can significantly improve the acoustic model. Overall, an absolute reduction of $1.1\%$ in WER on the development set and $0.9\%$ on the test set are achieved using the augmented acoustic model. These results are significant with $p < 0.001$ using the sign test calculated at the utterance level.

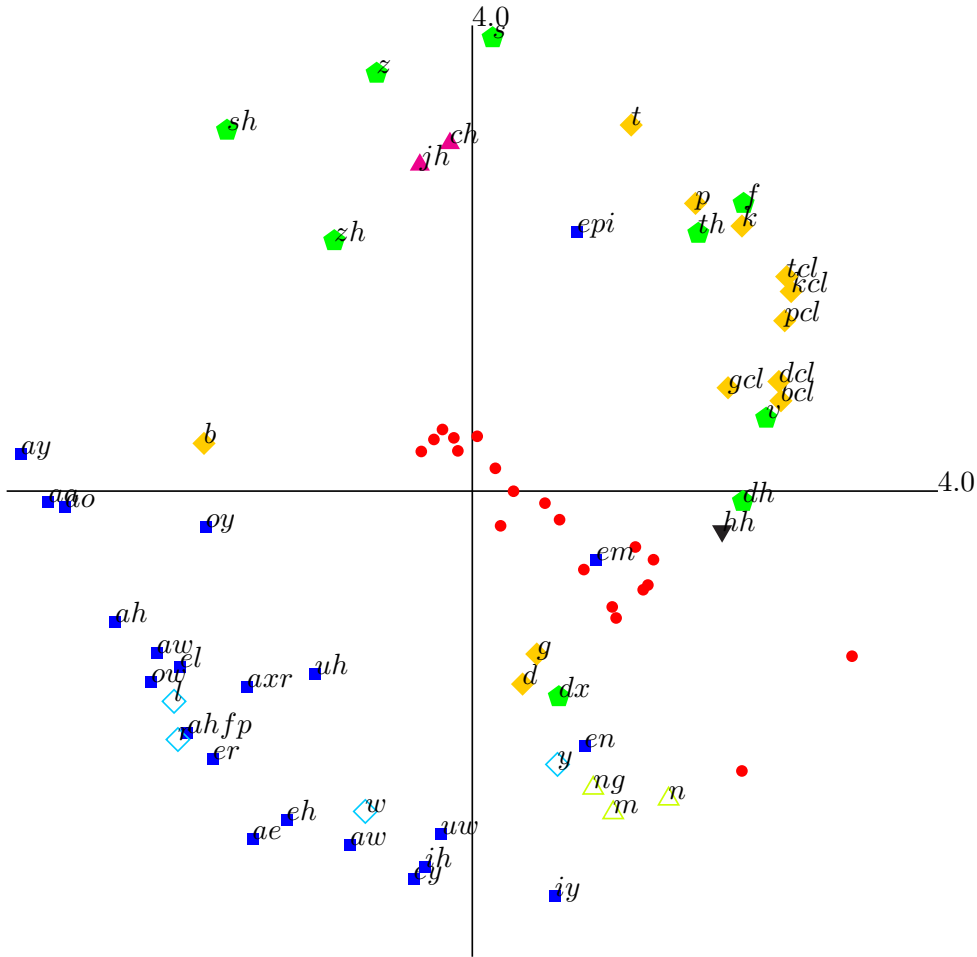

Figure 1: Plot of 2-dimensional output codes corresponding to 73 acoustic phonetic classes. The red circles indicate noise and silence classes. The phonemic classes are divided as follows: vowels, semivowels, nasals, stops and stop closures, fricatives, affricates, and the aspirant /hh/.

# 7 Discussion

## 7.1 Plot of a low-dimensional embedding

In order to get a sense of what is learned by the output codes of $P_{ecoc}$ we can plot the output codes directly. Figure 1 shows a plot of the output codes learned when $L = 2$. The output codes are learned for 1871 classes, but only 73 internal acoustic-phonetic classes are shown in the plot for clarity. In the plot, classes of similar acoustic-phonetic category are shown in the same color and shape. We can see that items of similar acoustic categories are grouped closely together. For example, the vowels are close to each other in the bottom left quadrant, while the stop-closures are grouped together in the top right, the affricates in the top left, and the nasals in the bottom right. The fricatives are a little more spread out but usually grouped close to another fricative that shares some underlying phonological feature such as /sh/ and /zh/ which are both palatal and /f/ and /th/ which are both unvoiced. We can also see specific acoustic properties emerging. For example the voiced stops /b/, /d/, /g/ are placed close to other voiced items of different acoustic categories.

## 7.2 Extensions

The ECOC embedding of the label space could also be co-learned with an embedding of the input acoustic vector space by extending the approach of NCA [8]. It would simply require the reintroduction of the projection matrix $\mathbf{A}$ in the weights $\alpha$.

$$\alpha(j|\mathbf{x}) = \frac{e^{-||\mathbf{A}\mathbf{x} - \mathbf{A}\mathbf{x}_j||^2}}{\sum_{m=1}^{N} e^{-||\mathbf{A}\mathbf{x} - \mathbf{A}\mathbf{x}_m||^2}}$$

$H(\mathbf{x}; \mathbf{M})$ and $P_{ecoc}$ would still be defined as in section 4.1. The optimization criterion would now depend on both $\mathbf{A}$ and $\mathbf{M}$. To optimize $\mathbf{A}$, we could again use gradient methods. Co-learning the two embeddings $\mathbf{M}$ and $\mathbf{A}$ could potentially lead to further improvements.

## 8 Conclusion

We have shown that nearest neighbor methods can be used to improve the performance of a GMM-based acoustic model and reduce the WER on a challenging speech recognition task. We have also developed a model for using error-correcting output codes to represent an embedding of the acoustic-phonetic label space that helps us capture cross-class information. Future work on this task could include co-learning an embedding of the input acoustic vector space with the ECOC matrix to attempt to achieve further gains.

## Appendix

We define three distributions based on the prior probabilities, $P(y)$, of the acoustic phonetic classes. The SUMMIT recognizer makes use of 1871 distinct acoustic phonetic labels [5]. We divide the set of labels, $\mathcal{Y}$, into three disjoint categories.

- $\mathcal{Y}^{(1)}$ includes labels involving *internal* phonemic events (e.g. /ay/)
- $\mathcal{Y}^{(2)}$ includes labels involving the *transition* from one acoustic-phonetic event to another (e.g. /ow/->/ch/)
- $\mathcal{Y}^{(3)}$ includes labels involving only *non-phonetic* events like noise and silence

We define a distribution $P^{(1)}(y)$ as follows. Distributions $P^{(2)}(y)$ and $P^{(3)}(y)$ are defined similarly.

$$P^{(1)}(y) = \frac{\begin{cases} P(y), & \text{if } y \in \mathcal{Y}^{(1)} \\ 0, & \text{otherwise} \end{cases}}{\sum_{y' \in \mathcal{Y}^{(1)}} P(y')}$$

## References

[1] E. L. Allwein, R. E. Schapire, and Y. Singer. Reducing multiclass to binary: a unifying approach for margin classifiers. *Journal of Machine Learning Research*, 1:113–141, 2000.

[2] K. Crammer and Y. Singer. Improved output coding for classification using continuous relaxation. In *Advances in Neural Information Processing Systems*. MIT Press, 2000.

[3] K. Crammer and Y. Singer. On the learnability and design of output codes for multiclass problems. *Machine Learning*, 47(2-3):201–233, 2002.

[4] T. G. Dietterich and G. Bakiri. Solving multiclass learning problems via error-correcting output codes. *Journal of Artificial Intelligence Research*, 2:263–286, 1995.

[5] J. Glass. A probabilistic framework for segment-based speech recognition. *Computer, Speech, and Language*, 17(2-3):137–152, 2003.

[6] J. Glass, T. J. Hazen, L. Hetherington, and C. Wang. Analysis and processing of lecture audio data: Preliminary investigations. In *HLT-NAACL 2004 Workshop on Interdisciplinary Approaches to Speech Indexing and Retrieval*, pages 9–12, 2004.

[7] A. Globerson and S. Roweis. Metric learning by collapsing classes. In Y. Weiss, B. Scholkopf, and J. Platt, editors, *Advances in Neural Information Processing Systems 18*, pages 513–520. MIT Press, 2006.

[8] J. Goldberger, S. Roweis, G. Hinton, and R. Salakhutdinov. Neighbourhood components analysis. In L. K. Saul, Y. Weiss, and L. Bottou, editors, *Advances in Neural Information Processing Systems 17*, pages 513–520. MIT Press, 2005.

[9] A. Klautau, N. Jevtic, and A. Orlitsky. On nearest-neighbor error-correcting output codes with aplication to all-pairs multiclass support vector machines. *Journal of Machine Learning Research*, 4:1–15, 2003.

[10] W. H. Press, S. A. Teukolsky, W. T. Vetterline, and B. P. Flannery. *Numerical recipes: the art of scientific computing*. Cambridge University Press, 3 edition, 2007.

[11] O. Pujol, P. Radeva, and J. Vitria. Discriminant ecoc: a heuristic method for application dependent design of error correcting output codes. *IEEE Transactions of Pattern Analysis and Machine Intelligence*, 28(6), 2006.

[12] R. Salakhutdinov and G. Hinton. Learning a nonlinear embedding by preserving class neighbourhood structure. *AI and Statistics*, 2007.

[13] N. Singh-Miller, M. Collins, and T. J. Hazen. Dimensionality reduction for speech recognition using neighborhood components analysis. In *Interspeech*, 2007.

[14] A. Torralba, R. Fergus, and Y. Weiss. Small codes and large image databases for recognition. *IEEE Computer Vision and Pattern Recognition*, June 2008.

[15] K. Q. Weinberger, J. Blitzer, and L. K. Saul. Distance metric learning for large margin nearest neighbor classification. In *Advances in Neural Information Processing Systems*. MIT Press, 2006.

[16] G. Zavaliagkos, Y. Zhao, R. Schwartz, and J. Makhoul. A hybrid segmental neural net/hidden markov model system for continuous speech recognition. *IEEE Transactions on Speech and Audio Processing*, 2(1):151–160, 1994.

